# Where does the population vector of motor cortical cells point during reaching movements?

**Pierre Baraduc***
pbaraduc@snv.jussieu.fr

**Emmanuel Guigon**
guigon@ccr.jussieu.fr

**Yves Burnod**
ybteam@ccr.jussieu.fr

INSERM U483, Université Pierre et Marie Curie
9 quai St Bernard, 75252 Paris cedex 05, France

## Abstract

Visually-guided arm reaching movements are produced by distributed neural networks within parietal and frontal regions of the cerebral cortex. Experimental data indicate that (1) single neurons in these regions are broadly tuned to parameters of movement; (2) appropriate commands are elaborated by populations of neurons; (3) the coordinated action of neurons can be visualized using a neuronal population vector (NPV). However, the NPV provides only a rough estimate of movement parameters (direction, velocity) and may even fail to reflect the parameters of movement when arm posture is changed. We designed a model of the cortical motor command to investigate the relation between the desired direction of the movement, the actual direction of movement and the direction of the NPV in motor cortex. The model is a two-layer self-organizing neural network which combines broadly-tuned (muscular) proprioceptive and (cartesian) visual information to calculate (angular) motor commands for the initial part of the movement of a two-link arm. The network was trained by motor babbling in 5 positions. Simulations showed that (1) the network produced appropriate movement direction over a large part of the workspace; (2) small deviations of the actual trajectory from the desired trajectory existed at the extremities of the workspace; (3) these deviations were accompanied by large deviations of the NPV from both trajectories. These results suggest the NPV does not give a faithful image of cortical processing during arm reaching movements.

# 1   INTRODUCTION

When reaching to an object, our brain transforms a visual stimulus on the retina into a finely coordinated motor act. This complex process is subserved in part by distributed neuronal populations within parietal and frontal regions of the cerebral cortex (Kalaska and Crammond 1992). Neurons in these areas contribute to coordinate transformations by encoding target position and kinematic parameters of reaching movements in multiple frames of reference and to the elaboration of motor commands by sending directional and positional signals to the spinal cord (Georgopoulos 1996). An ubiquitous feature of cortical populations is that most neurons are broadly tuned to a preferred attribute (e.g. direction) and that tuning curves are uniformly (or regularly) distributed in the attribute space (Georgopoulos 1996). Accordingly, a powerful tool to analyse cortical populations is the NPV which describes the behavior of a whole population by a single vector (Georgopoulos 1996). Georgopoulos et al. (1986) have shown that the NPV calculated on a set of directionally tuned neurons in motor cortex points approximately (error $\sim 15°$) in the direction of movement. However, the NPV may fail to indicate the correct direction of movement when the arm is in a particular posture (Scott and Kalaska 1995). These data raise two important questions: (1) how populations of broadly tuned neurons learn to compute a correct sensorimotor transformation? Previous models (Burnod et al. 1992; Bullock et al. 1993; Salinas and Abbott 1995) provided partial solutions to this problem but we still lack a model which closely matches physiological and psychophysical data on reaching movements; (2) Are cortical processes involved in the visual guidance of arm movements readable with the NPV tool? This article provides answers to these questions through a physiologically inspired model of sensorimotor transformations.

# 2   MODEL OF THE VISUAL-TO-MOTOR TRANSFORMATION

## 2.1   ARM GEOMETRY

The arm model has voluntarily been chosen simple. It is a planar, two-link arm, with limited (160 degrees) joint excursion at shoulder and elbow. An agonist/antagonist pair is attached at each joint.

## 2.2   INPUT AND OUTPUT CODINGS

No cell is finely tuned to a specific input or output value to mimic the broad tunings or monotonic firing characteristics found in cortical visuomotor areas.

### 2.2.1   Arm position

By analogy with the role of muscle spindles, proprioceptive sensors are assumed to code muscle length. Arm position is thus represented by the population activity of $N_r = 20$ neurons coding for the length of each agonist or antagonist. The activity of a sensor neuron $k$ is defined by:

$$\tau_k = \sigma_k(L_{n(k)})$$

where $L_{n(k)}$ is the length of muscle number $n(k)$, and $\sigma_k$ a piecewise linear sigmoid:

$$\sigma_k(L) = \begin{cases} 0 & : \quad L \leq \lambda_k \\ (L - \lambda_k)/(\Lambda_k - \lambda_k) & : \quad \lambda_k < L < \Lambda_k \\ 1 & : \quad L \geq \Lambda_k \end{cases}$$

Sensibility thresholds $\lambda_k$ are uniformly distributed in $[L_{min}, L_{max}]$, and the dynamic range is $\Lambda_k - \lambda_k$ is taken constant, equal to $L_{max} - L_{min}$.

### 2.2.2 Desired direction

The direction **V** of the desired movement in visual space is coded by a population of $N_x = 50$ neurons with cosine tuning in cartesian space. Each visual neuron $j$ thus fires as:

$$x_j = \mathbf{V} \cdot \mathbf{V}_j$$

$\mathbf{V}_j$ being the preferred direction of the cell. These 50 preferred directions are chosen uniformly distributed in 2-D space.

### 2.2.3 Motor Command

In attempt to model the existence of muscular synergies (Lemon 1988), we identified motor command with joint movement rather than with muscle contraction. A motor neuron $i$ among $N_t = 50$ contributes to the effective movement **M** by its action on a synergy (direction in joint space) $\mathbf{M}_i$. This collective effect is formally expressed by:

$$\mathbf{M} = \sum_i t_i \mathbf{M}_i$$

where $t_i$ is the activity of motor neuron $i$. The 50 directions of action $\mathbf{M}_i$ are supposed uniformly distributed in joint space.

## 3 NETWORK STRUCTURE AND LEARNING

### 3.1 STRUCTURE OF THE NETWORK

Information concerning the position of the arm and the desired direction in cartesian space

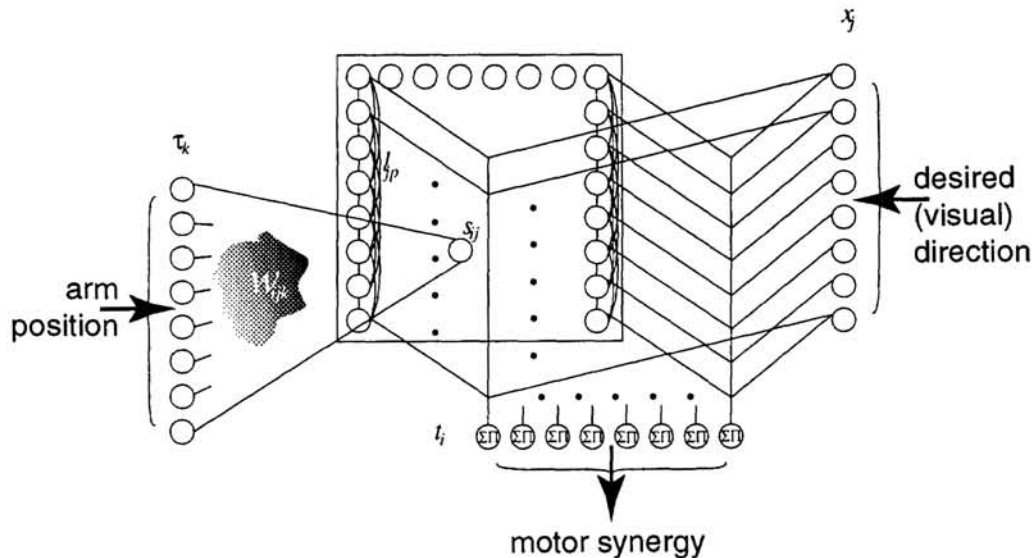

Figure 1: Network Architecture

is combined asymmetrically (Fig. 1). First, an intermediate (somatic) layer of neurons

forms an internal representation of the arm position by a combination of the input from the $N_\tau$ muscle sensors and the lateral interactions inside the population. Activity in this layer is expressed by:

$$s_{ij} = \sum_k w_{ijk}\, \tau_k + \sum_p l_{jp}\, s_{ip} \qquad (1)$$

where the lateral connections are:

$$l_{jp} = \cos\left(2\pi(j - p)/N_\tau\right)$$

Equation 1 is self-referent; so calculation is done in two steps. The feed-forward input first arrives at time zero when there is no activity in the layer; iterated action of the lateral connections comes into play when this feed-forward input vanishes.

The activity in the somatic layer is then combined with the visual directional information by the output sigma-pi neurons as follows:

$$t_i = \sum_j x_j\, s_{ij}$$

## 3.2   WEIGHTS AND LEARNING

The only adjustable weights are the $w_{ijk}$ linking the proprioceptive layer to the somatic layer. Connectivity is random and not complete: only 15% of the somatic neurons receive information on arm position. The visuomotor mapping is learnt by modifying the internal representation of the arm.

Motor commands issued by the network are correlated with the visual effect of the movement ("motor babbling"). More precisely, the learning algorithm is a repetition of the following cycle:

1.  choice of an arm position among 5 positions (stars on Fig. 2)
2.  random emission of a motor command ($t_i$)
3.  corresponding visual reafference ($x_j$)
4.  weight modification according to a variant of the delta rule:
    $$\Delta w_{ijk} \propto (t_i x_j - s_{ij})\, \tau_k$$

The random commands are gaussian distributions of activity over the output layer. 5000 learning epochs are sufficient to obtain a stabilized performance. It must be noted that the error between the ideal response of the network and the actual performance never decreases completely to zero, as the constraints of the visuomotor transformation vary over the workspace.

# 4   RESULTS

## 4.1   NETWORK PERFORMANCE

Correct learning of the mapping was tested in 21 positions in the workspace in a pointing task toward 16 uniformly distributed directions in cartesian space. Movement directions generated by the network are shown in Fig. 2 (desired direction 0 degree is shown bold). Norm of movement vectors depends on the global activity in the network which varies with arm position and movement direction.

Performance of the network is maximal near the learning positions. However, a good generalization is obtained (directional error 0.3°, SD 12.1°); a bias toward the shoulder can be observed in extreme right or left positions. A similar effect was observed in psychophysical experiments (Ghilardi et al. 1995).

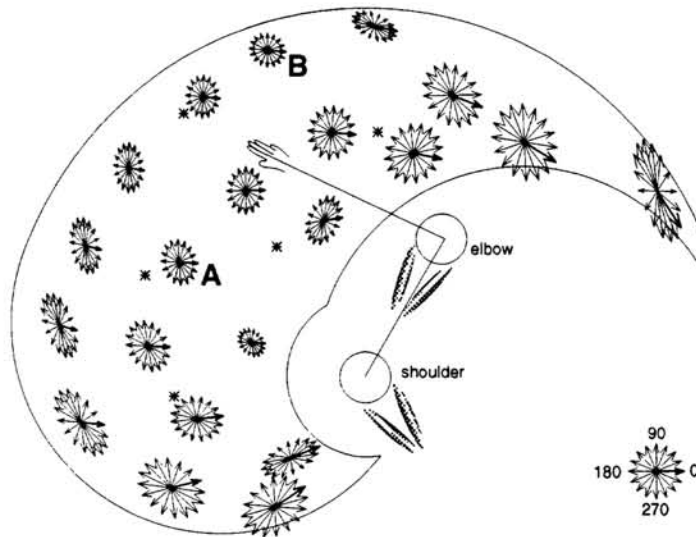

Figure 2: Performance in a pointing task

## 4.2 PREFERRED DIRECTIONS AND POPULATION VECTOR

### 4.2.1 Behavior of the population vector

Preferred directions (PD) of output units were computed using a multilinear regression; a perfect cosine tuning was found, which is a consequence of the *exact* multiplication in sigma-pi neurons. Then, the population vector, the effective movement vector, and the desired movement were compared (Fig. 3) for two different arm configurations A and B marked on Fig. 2. The movement generated by the network (dashed arrow) is close to the

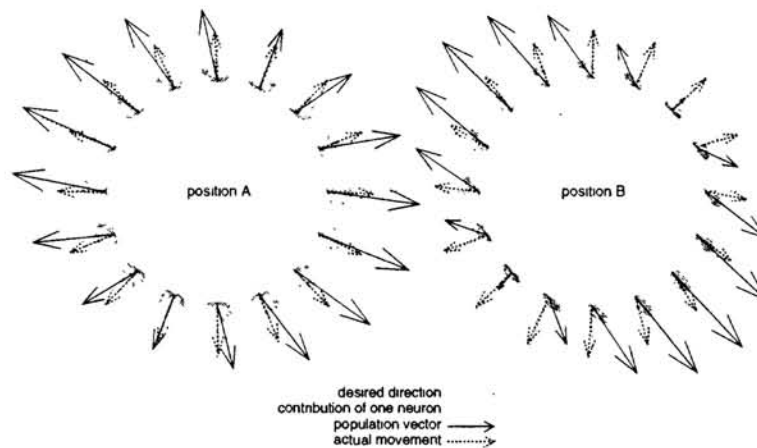

Figure 3: Actual movement and population vector in two arm positions

desired one (dotted rays) for both arm configurations. However, the population vector (solid arrow) is not always aligned with the movement. The discrepancy between movement and population vector depends both on the direction and the position of the arm: it is maximal

for positions near the borders of the workspace as position B. Fig. 3 (position B) shows that the deviations of the population vector are due to the anisotropic distribution of the PDs in cartesian space for given positions.

### 4.2.2   Difference between direction of action and preferred direction

Marked anisotropy in the distribution of PDs is compatible with accurate performance. To see why, let us call "direction of action" (DA) the motor cell's contribution to the movement. The distribution of DAs presents an anisotropy due to the geometry of the arm. This anisotropy is canceled by the distribution of PDs. Mathematically, if $U$ is a $N \times 2$ matrix of uniformly distributed 2D vectors, the PD matrix is $UJ^{-1}$ whereas the DA matrix is $UJ^{T}$, $J$ being the jacobian of the angular-to-cartesian mapping. Difference between DA and PD has been plotted with concentric arcs for four representative neurons at 21 arm positions in Fig. 4. Sign and magnitude of the difference vary continuously over the workspace and

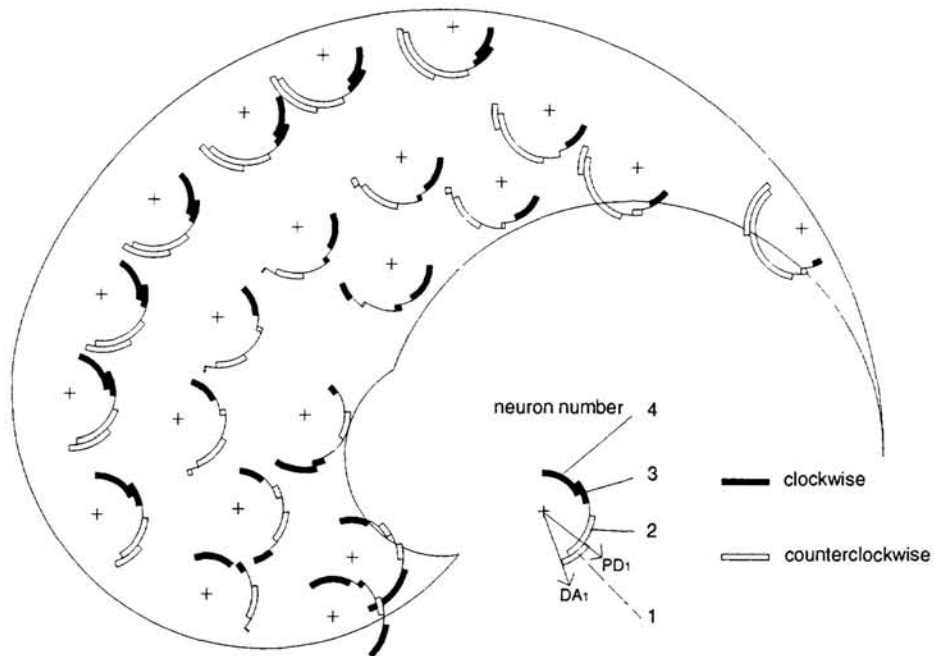

Figure 4: Difference between direction of action and preferred direction for four units.

often exceed 45 degrees. It can also be noted that preferred directions rotate with the arm as was experimentally noted by (Caminiti et al. 1991).

## 5   DISCUSSION

We first asked how a network of broadly tuned neurons could produce visually guided arm movements. The model proposed here produces a correct behavior over the entire workspace. Biases were observed at the extreme right and left which closely resemble experimental data in humans (Ghilardi et al. 1995). Single cells in the output layer behave as motor cortical cells do and the NPV of these cells correctly indicated the direction of movement for hand positions in the central region of the workspace (see Caminiti et al. 1991). Models of sensorimotor transformations have already been proposed. However they either considered motor synergies in cartesian coordinates (Burnod et al. 1992), or used sharply

tuned units (Bullock et al. 1993), or motor effects independent of arm position (Salinas and Abbott 1995). Next, the use of the NPV to describe cortical activity was questioned. A fundamental assumption in the calculation of the NPV is that the PD of a neuron is the direction in which the arm would move if the neuron were stimulated. The model shows that the two directions DA and PD do not necessarily coincide, which is probably the case in motor cortex (Scott and Kalaska 1995). It follows that the NPV often points neither in the actual movement direction nor in the desired movement direction (target direction), especially for unusual arm configurations. A maximum-likelihood estimator does not have these flaws; it would however accurately predict the *desired* movement out of the *output* unit activities, even for a wrong actual movement. In conclusion: (1) the NPV does not provide a faithful image of cortical visuomotor processes; (2) a correct NPV should be based on the DAs, which cannot easily be determined experimentally; (3) planning of trajectories in space cannot be realized by the successive recruitment of motor neurons whose PDs sequentially describe the movement.

## Footnotes

*to whom correspondence should be addressed

## References

Bullock, D., S. Grossberg, and F. Guenther (1993). A self-organizing neural model of motor equivalent reaching and tool use by a multijoint arm. *J Cogn Neurosci 5*(4), 408–435.

Burnod, Y., P. Grandguillaume, I. Otto, S. Ferraina, P. Johnson, and R. Caminiti (1992). Visuomotor transformations underlying arm movements toward visual targets: a neural network model of cerebral cortical operations. *J Neurosci 12*(4), 1435–53.

Caminiti, R., P. Johnson, C. Galli, S. Ferraina, and Y. Burnod (1991). Making arm movements within different parts of space: the premotor and motor cortical representation of a coordinate system for reaching to visual targets. *J Neurosci 11*(5), 1182–97.

Georgopoulos, A. (1996). On the translation of directional motor cortical commands to activation of muscles via spinal interneuronal systems. *Brain Res Cogn Brain Res 3*(2), 151–5.

Georgopoulos, A., A. Schwartz, and R. Kettner (1986). Neuronal population coding of movement direction. *Science 233*(4771), 1416–9.

Ghilardi, M., J. Gordon, and C. Ghez (1995). Learning a visuomotor transformation in a local area of work space pr oduces directional biases in other areas. *J Neurophysiol 73*(6), 2535–9.

Kalaska, J. and D. Crammond (1992). Cerebral cortical mechanisms of reaching movements. *Science 255*(5051), 1517–23.

Lemon, R. (1988). The output map of the primate motor cortex. *Trends Neurosci 11*(11), 501–6.

Salinas, E. and L. Abbott (1995). Transfer of coded information from sensory to motor networks. *J Neurosci 15*(10), 6461–74.

Scott, S. and J. Kalaska (1995). Changes in motor cortex activity during reaching movements with similar hand paths but different arm postures. *J Neurophysiol 73*(6), 2563–7.
